# Generalization Performance in PARSEC—A Structured Connectionist Parsing Architecture

Ajay N. Jain*
School of Computer Science
Carnegie Mellon University
Pittsburgh, PA 15213-3890

## ABSTRACT

This paper presents PARSEC—a system for generating connectionist parsing networks from example parses. PARSEC is not based on formal grammar systems and is geared toward spoken language tasks. PARSEC networks exhibit three strengths important for application to speech processing: 1) they *learn* to parse, and generalize well compared to hand-coded grammars; 2) they tolerate several types of noise; 3) they can learn to use multi-modal input. Presented are the PARSEC architecture and performance analyses along several dimensions that demonstrate PARSEC's features. PARSEC's performance is compared to that of traditional grammar-based parsing systems.

## 1 INTRODUCTION

While a great deal of research has been done developing parsers for natural language, adequate solutions for some of the particular problems involved in spoken language have not been found. Among the unsolved problems are the difficulty in constructing task-specific grammars, lack of tolerance to noisy input, and inability to effectively utilize non-symbolic information. This paper describes PARSEC—a system for generating connectionist parsing networks from example parses.

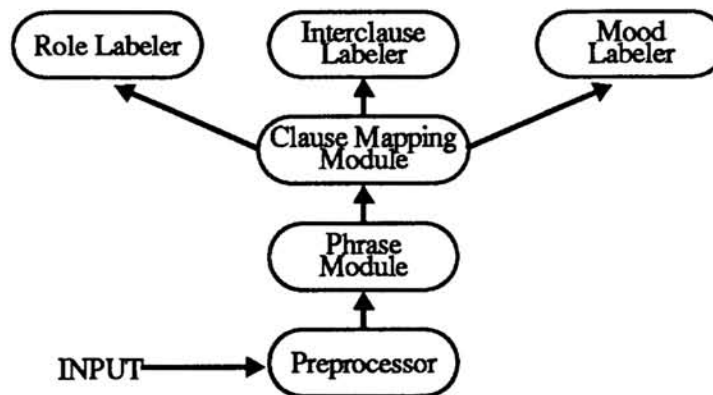

Figure 1: PARSEC's high-level architecture

PARSEC networks exhibit three strengths:

- They automatically learn to parse, and generalize well compared to hand-coded grammars.
- They tolerate several types of noise without any explicit noise modeling.
- They can learn to use multi-modal input such as pitch in conjunction with syntax and semantics.

The PARSEC network architecture relies on a variation of supervised back-propagation learning. The architecture differs from some other connectionist approaches in that it is highly structured, both at the macroscopic level of modules, and at the microscopic level of connections. Structure is exploited to enhance system performance.[1]

Conference registration dialogs formed the primary development testbed for PARSEC. A separate speech recognition effort in conference registration provided data for evaluating noise-tolerance and also provided an application for PARSEC in speech-to-speech translation (Waibel *et al.* 1991).

PARSEC differs from early connectionist work in parsing (e.g. Fanty 1985; Selman 1985) in its emphasis on learning. It differs from recent connectionist approaches (e.g. Elman 1990; Miikkulainen 1990) in its emphasis on performance issues such as generalization and noise tolerance in real tasks. This papers presents the PARSEC architecture, its training algorithms, and performance analyses that demonstrate PARSEC's features.

## 2   PARSEC ARCHITECTURE

The PARSEC architecture is modular and hierarchical. Figure 1 shows the high-level architecture. PARSEC can learn to parse complex English sentences including multiple clauses, passive constructions, center-embedded constructions, etc. The input to PARSEC is presented sequentially, one word at a time. PARSEC produces a case-based representation of a parse as the input sentence develops.

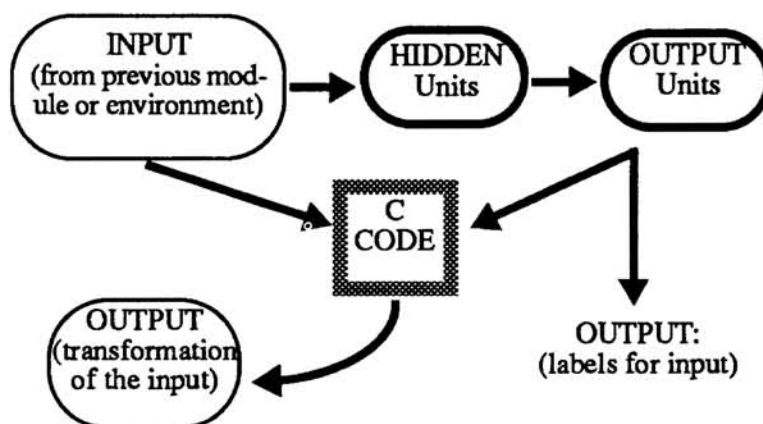

Figure 2: Basic structure of a PARSEC module

The parse for the sentence, "I will send you a form immediately," is:

```
([statement]
    ([clause]
        ([agent]      I)
        ([action]     will send)
        ([recipient]  you)
        ([patient]    a form)
        ([time]       immediately)))
```

Input words are represented as binary feature patterns (primarily syntactic with some semantic features). These feature representations are hand-crafted.

Each module of PARSEC can perform either a transformation or a labeling of its input. The output function of each module is represented across localist connectionist units. The actual transformations are made using non-connectionist subroutines.[2] Figure 2 shows the basic structure of a PARSEC module. The bold ovals contain units that learn via back-propagation.

There are four steps in generating a PARSEC network: 1) create an example parse file; 2) define a lexicon; 3) train the six modules; 4) assemble the full network. Of these, only the first two steps require substantial human effort, and this effort is small relative to that required for writing a grammar by hand. Training and assembly are automatic.

## 2.1 PREPROCESSING MODULE

This module marks alphanumeric sequences, which are replaced by a single special marker word. This prevents long alphanumeric strings from overwhelming the length constraint on phrases. Note that this is not always a trivial task since words such as "a" and "one" are lexically ambiguous.

INPUT:    "It costs three hundred twenty one dollars."
OUTPUT:   "It costs ALPHANUM dollars."

## 2.2 PHRASE MODULE

The Phrase module processes the evolving output of the Prep module into *phrase blocks*. Phrase blocks are non-recursive contiguous pieces of a sentence. They correspond to simple noun phrases and verb groups.[3] Phrase blocks are represented as grouped sets of units in the network. Phrase blocks are denoted by brackets in the following:

INPUT:     "I will send you a new form in the morning."
OUTPUT:   "[I] [will send] [you] [a new form] [in the morning]."

## 2.3 CLAUSE MAPPING MODULE

The Clause module uses the output of the Phrase module as input and assigns the clausal structure. The result is an unambiguous bracketing of the phrase blocks that is used to transform the phrase block representation into representations for each clause:

INPUT:     "[I] [would like] [to register] [for the conference]."
OUTPUT:   "{[I] [would like]} {[to register] [for the conference]}."

## 2.4 ROLE LABELING MODULE

The Roles module associates case-role labels with each phrase block in each clause. It also denotes attachment structure for prepositional phrases ("MOD-1" indicates that the current phrase block modifies the previous one):

INPUT:     "{[The titles] [of papers] [are printed] [in the forms]}"
OUTPUT:   "{[The titles] [of papers] [are printed] [in the forms]}"
                 PATIENT    MOD-1    ACTION    LOCATION

## 2.5 INTERCLAUSE AND MOOD MODULES

The Interclause and Mood modules are similar to the Roles module. They both assign labels to constituents, except they operate at higher levels. The Interclause module indicates, for example, subordinate and relative clause relationships. The Mood module indicates the overall sentence mood (declarative or interrogative in the networks discussed here).

# 3   GENERALIZATION

Generalization in large connectionist networks is a critical issue. This is especially the case when training data is limited. For the experiments reported here, the training data was limited to twelve conference registration dialogs containing approximately 240 sentences with a vocabulary of about 400 words. Despite the small corpus, a large number of English constructs were covered (including passives, conditional constructions, center-embedded relative clauses, etc.).

A set of 117 disjoint sentences was obtained to test coverage. The sentences were generated by a group of people different from those that developed the 12 dialogs. These sentences used the same vocabulary as the 12 dialogs.

## 3.1 EARLY PARSEC VERSIONS

Straightforward training of a PARSEC network resulted in poor generalization performance, with only 16% of the test sentences being parsed correctly. One of the primary sources for error was positional sensitivity acquired during training of the three transformational modules. In the Phrase module, for example, each of the phrase boundary detector units was supposed to learn to indicate a boundary between words in specific positions.

Each of the units of the Phrase module is performing essentially the same job, but the network doesn't "know" this and cannot learn this from a small sample set. By sharing the connection weights across positions, the network is forced to be position insensitive (similar to TDNN's as in Waibel *et al.* 1989). After modifying PARSEC to use shared weights and localized connectivity in the lower three modules, generalization performance increased to 27%. The primary source of error shifted to the Roles module.

Part of the problem could be ascribed to the representation of phrase blocks. They were represented across rows of units that each define a word. In the phrase block "the big dog," "dog" would have appeared in row 3. This changes to row 2 if the phrase block is just "the dog." A network had to learn to respond to the heads of phrase blocks even though they moved around. An augmented phrase block representation in which the last word of the phrase block was copied to position 0 solved this problem. With the augmented phrase block representation coupled with the previous improvements, PARSEC achieved 44% coverage.

## 3.2 PARSEC: FINAL VERSION

The final version of PARSEC uses all of the previous enhancements plus a technique called *Programmed Constructive Learning* (PCL). In PCL, hidden units are added to a network one at a time as they are needed. Also, there is a specific series of hidden unit types for each module of a PARSEC network. The hidden unit types progress from being highly local in input connectivity to being more broad. This forces the networks to learn general predicates before specializing and using possibly unreliable information.

The final version of PARSEC was used to generate another parsing network.[4] Its performance was 67% (78% including near-misses). Table 1 summarizes these results.

## 3.3 COMPARISON TO HAND-CODED GRAMMARS

PARSEC's performance was compared to that of three independently constructed grammars. Two of the grammars were commissioned as part of a contest where the first prize ($700) went to the grammar-writer with best coverage of the test set and the second prize ($300) went to the other grammar writer.[5] The third grammar was independently constructed as part of the JANUS system (described later). The contest grammars achieved 25% and 38% coverage, and the other grammar achieved just 5% coverage of the test set

Table 1: PARSEC's comparative performance

| | Coverage | Noise | Ungram. |
|---|---|---|---|
| PARSEC V4 | 67% (78%) | 77% | 66% |
| Grammar 1 | 38% (39%) | – | 34% |
| Grammar 2 | 25% (26%) | – | 38% |
| Grammar 3 | 5% (5%) | 70% | 2% |

(see Table 1). All of the hand-coded grammars produced NIL parses for the majority of test sentences. In the table, numbers in parentheses include near-misses.

PARSEC's performance was substantially better than the best of the hand-coded grammars. PARSEC has a systematic advantage in that it is trained on the *incremental* parsing task and is exposed to partial sentences during training. Also, PARSEC's constructive learning approach coupled with weight sharing emphasizes local constraints wherever possible, and distant variations in input structure do not adversely affect parsing.

## 4   NOISE TOLERANCE

The second area of performance analysis for PARSEC was noise tolerance. Preliminary comparisons between PARSEC and a rule-based parser in the JANUS speech-to-speech translation system were promising (Waibel *et al.* 1991). More extensive evaluations corroborated the early observations. In addition, PARSEC was evaluated on synthetic ungrammatical sentences. Experiments on spontaneous speech using DARPA's ATIS task are ongoing.

### 4.1   NOISE IN SPEECH-TO-SPEECH TRANSLATION

In the JANUS system, speech recognition is provided by an LPNN (Tebelskis *et al.* 1991), parsing can be done by a PARSEC network or an LR parser, translation is accomplished by processing the interlingual output of the parser using a standard language generation module, and speech generation is provided by off-the-shelf devices. The system can be run using a single (often noisy) hypothesis from the LPNN or a ranked list of hypotheses.

When run in single-hypothesis mode, JANUS using PARSEC correctly translated 77% of the input utterances, and JANUS using the LR parser (Grammar 3 in the table) achieved 70%. The PARSEC network was able to parse a number of incorrect recognitions well enough that a successful translation resulted. However, when run in multi-hypothesis mode, the LR parser achieved 86% compared to PARSEC's 80%. The LR parser utilized a very tight grammar and was able to robustly reject hypotheses that deviated from expectations. This allowed the LR parser to "choose" the correct hypothesis more often than PARSEC. PARSEC tended to accept noisy utterances that produced incorrect translations. Of course, given that the PARSEC network's coverage was so much higher than that of the grammar used by the LR parser, this result is not surprising.

### 4.2   SYNTHETIC UNGRAMMATICALITY

Using the same set of grammars for comparison, the parsers were tested on ungrammatical input from the CR task. These sentences were corrupted versions of sentences used for

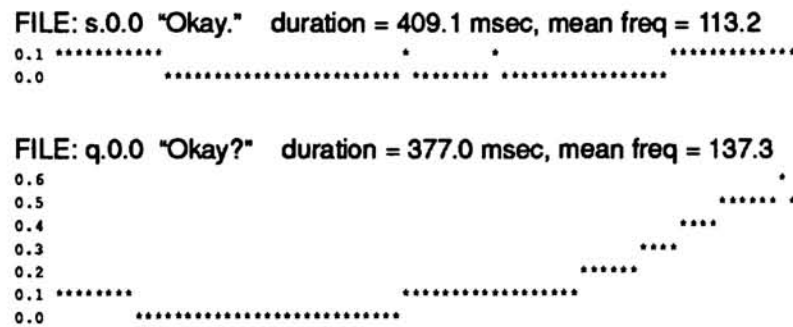

Figure 3: Smoothed pitch contours.

*training*. Training sentences were used to decouple the effects of noise from coverage. Table 1 shows the results. They essentially mirror those of the coverage tests. PARSEC is substantially less sensitive to such effects as subject/verb disagreement, missing determiners, and other non-catastrophic irregularities.

Some researchers have augmented grammar-based systems to be more tolerant of noise (e.g. Saito and Tomita 1988). However, the PARSEC network in the test reported here was trained only on grammatical input and still produced a degree of noise tolerance for free. In the same way that one can explicitly build noise tolerance into a grammar-based system, one can train a PARSEC network on input that includes specific types of noise. The result should be some noise tolerance beyond what was explicitly trained.

## 5  MULTI-MODAL INPUT

A somewhat elusive goal of spoken language processing has been to utilize information from the speech signal beyond just word sequences in higher-level processing. It is well known that humans use such information extensively in conversation. Consider the utterances "Okay." and "Okay?" Although semantically distinct, they cannot be distinguished based on word sequence, but pitch contours contain the necessary information (Figure 3).

In a grammar-based system, it is difficult to incorporate real-valued vector input in a useful way. In a PARSEC network, the vector is just another set of input units. The Mood module of a PARSEC network was augmented to contain an additional set of units that contained pitch information. The pitch contours were smoothed output from the OGI Neural Network Pitch Tracker (Barnard *et al.* 1991). PARSEC added another hidden unit to utilize the new information.

The trained PARSEC network was tolerant of speaker variation, gender variation, utterance variation (length and content), and a combination of these factors. Although not explicitly trained to do so, the network correctly processed sentences that were grammatical questions but had been pronounced with the declining pitch of a typical statement.

Within the JANUS system, the augmented PARSEC network brings new functionality. Intonation affects translation in JANUS when using the augmented PARSEC network. The sentence, "This is the conference office." is translated to "Kaigi jimukyoku desu." "This is the conference office?" is translated to "Kaigi jimukyoku desuka?" This required no changes in the other modules of the JANUS system. It also should be possible to use other types of information from the speech signal to aid in robust parsing (e.g. energy patterns to disambiguate clausal structure).

# 6  CONCLUSION

PARSEC is a system for generating connectionist parsing networks from training examples. Experiments using a conference registration conversational task showed that PARSEC: 1) learns and generalizes well compared to hand-coded grammars; 2) tolerates noise: recognition errors and ungrammaticality; 3) successfully learns to combine intonational information with syntactic/semantic information. Future work with PARSEC will be continued by extending it to new languages, larger English tasks, and speech tasks that involve tighter coupling between speech recognition and parsing. There are numerous issues in NLP that will be addressed in the context of these research directions.

## Acknowledgements

The author gratefully acknowledges the support of DARPA, the National Science Foundation, ATR Interpreting Telephony Laboratories, NEC Corp., and Siemens Corp.

## Footnotes

*Now with Alliant Techsystems Research and Technology Center (jain@rtc.atk.com).

[1]PARSEC is a generalization of a previous connectionist parsing architecture (Jain 1991). For a detailed exposition of PARSEC, please refer to Jain's PhD thesis (in preparation).

[2] These transformations could be carried out by connectionist networks, but at a substantial computational cost for training and a risk of undergeneralization.

[3]Abney has described a similar linguistic unit called a *chunk* (Abney 1991).

[4]This final parsing network was not trained all the way to completion. Training to completion hurts generalization performance.

[5]The contest participants had 8 weeks to complete their grammars, and they both spent over 60 hours doing so. The grammar writers work in Machine Translation and Computational Linguistics and were quite experienced.

## References

Abney, S. P. 1991. Parsing by chunks. In *Principle-Based Parsing*, ed. R. Berwick, S. P. Abney, C. Tenny. Kluwer Academic Publishers.

Barnard, E., R. A. Cole, M. P. Vea, F. A. Alleva. 1991. Pitch Detection with a Neural-Net Classifier. *IEEE Transactions on Signal Processing* 39(2): 298–307.

Elman, J. L. 1989. *Representation and Structure in Connectionist Networks*. Tech. Rep. CRL 8903. Center for Research in Language, University of California, San Diego.

Fanty, M. 1985. *Context Free Parsing in Connectionist Networks*. Tech. Rep. TR174, Computer Science Department, University of Rochester.

Jain, A. N. and A. H. Waibel. 1990. Robust connectionist parsing of spoken language. In *Proceedings of the 1990 IEEE International Conference on Acoustics, Speech, and Signal Processing*

Jain, A. N. In preparation. *PARSEC: A Connectionist Learning Architecture for Parsing Speech*. PhD Thesis. School of Computer Science, Carnegie Mellon University.

Miikkulainen, R. 1990. A PDP architecture for processing sentences with relative clauses. In *Proceedings of the 13th Annual Conference of the Cognitive Science Society*.

Saito, H., and M. Tomita. 1988. Parsing noisy sentences. In *Proceedings of INFO JAPAN '88: International Conference of the Information Processing Society of Japan*, 553–59.

Selman, B. 1985. *Rule-Based Processing in a Connectionist System for Natural Language Understanding*. Ph.D. Thesis, University of Toronto. Available as Tech. Rep. CSRI-168.

Tebelskis, J., A. Waibel, B. Petek, and O. Schmidbauer. 1991. Continuous speech recognition using linked predictive neural networks. In *Proceedings of the 1991 IEEE International Conference on Acoustics, Speech, and Signal Processing*.

Waibel, A., T. Hanazawa, G. Hinton, K. Shikano, and K. Lang. 1989. Phoneme recognition using time-delay neural networks. *IEEE Transactions on Acoustics, Speech, and Signal Processing* 37(3):328–339.

Waibel, A., A. N. Jain, A. E. McNair, H. Saito, A. G. Hauptmann, and J. Tebelskis. 1991. JANUS: A speech-to-speech translation system using connectionist and symbolic processing strategies. In *IEEE Proceedings of the International Conference on Acoustics, Speech, and Signal Processing*.